# Online Pricing with Strategic and Patient Buyers

**Michal Feldman**
Tel-Aviv University and MSR Herzliya
michal.feldman@cs.tau.ac.il

**Tomer Koren**[*]
Google Brain
tkoren@google.com

**Roi Livni**[*]
Princeton University
rlivni@cs.princeton.edu

**Yishay Mansour**[*]
Tel-Aviv University
mansour@tau.ac.il

**Aviv Zohar**[*]
Hebrew University of Jerusalem
avivz@cs.huji.ac.il

## Abstract

We consider a seller with an unlimited supply of a single good, who is faced with a stream of $T$ buyers. Each buyer has a window of time in which she would like to purchase, and would buy at the lowest price in that window, provided that this price is lower than her private value (and otherwise, would not buy at all). In this setting, we give an algorithm that attains $O(T^{2/3})$ regret over any sequence of $T$ buyers with respect to the best fixed price in hindsight, and prove that no algorithm can perform better in the worst case.

## 1   Introduction

Perhaps the most common way to sell items is using a "posted price" mechanism in which the seller publishes the price of an item in advance, and buyers that wish to obtain the item decide whether to acquire it at the given price or to forgo the purchase. Such mechanisms are extremely appealing. The decision made by the buyer in a single-shot interaction is simple: if it values the item by more than the offering price, it should buy, and if its valuation is lower, it should decline. The seller on the other hand needs to determine the price at which she wishes to sell goods. In order to set prices, additive regret can be minimized using, for example, a multi-armed bandit (MAB) algorithm in which arms correspond to a different prices, and rewards correspond to the revenue obtained by the seller.

Things become much more complicated when the buyers who are facing the mechanism are *patient* and can choose to wait for the price to drop. The simplicity of posted price mechanisms is then tainted by strategic considerations, as buyers attempt to guess whether or not the seller will lower the price in the future. The direct application of MABs is no longer adequate, as prices set by such algorithms may fluctuate at every time period. Strategic buyers can make use of this fact to gain the item at a lower price, which lowers the revenue of the seller and, more crucially, changes the seller's feedback for a given price. With patient buyers, the revenue from sales is no longer a result of the price at the current period alone, but rather the combined outcome of prices that were set in surrounding time periods, and of the expectation of buyers regarding future prices.

In this paper, we focus on strategic buyers that may delay their purchase in hopes of obtaining a better deal. We assume that each buyer has a valuation for the item, and a "patience level" which represents the length of the time-window during which it is willing to wait in order to purchase the item. Buyers wish to minimize the price during this period. Note that such buyers may interfere with naïve attempts to minimize regret, as consecutive days at which different prices are set are no longer independent.

---

[*]Parts of this work were done while the author was at Microsoft Research, Herzliya.

To regain the simplicity of posted prices for the buyers, we consider a setting in which the seller commits to the price in subsequent time periods in advance, publishing prices for the entire window of the buyers. Strategic buyers that arrive at the market are then able to immediately choose the lowest price within their window. Thus, given the valuation and patience of the buyers (the number of days they are willing to wait) their actions are clearly determined: buy it at a day that is within the buyer's patience window and price is cheapest, provided that it is lower than the valuation.

An important aspect of our proposed model is to consider for each buyer a window of time (rather than, for example, discounting). For example, when considering discounting, the buyers, in order to best respond, would have argue how would other buyers would behave and how would the seller adjust the prices in response to them. By fixing a window of time, and forcing the seller to publish prices for the entire window, the buyers become "price takers" and their behavior becomes tractable to analyze.

As in previous works, we focus on minimizing the additive regret of the seller, assuming that the appearance of buyers is adversarial; that is, we do not make any statistical assumptions on the buyers' valuation and window size (except for a simple upper bound). Specifically we assume that the values are in the range $[0, 1]$ and that the window size is in the range $\{1, \ldots, \hat{\tau} + 1\}$. The regret is measured with respect to the best single price in hindsight. Note that the benchmark of a fixed price $p^*$ implies that any buyer with value above $p^*$ buys and any buyer with value below $p^*$ does not buy. The window size has no effect when we have a fixed price. On the other hand, for the online algorithm, having to deal with various window sizes create a new challenge.

The special case of this model where $\hat{\tau} = 0$ (and hence all buyers have window of size exactly one) was previously studied by Kleinberg and Leighton [11], who discussed a few different models for the buyer valuations and derived tight regret bounds for them. When the set of feasible prices is of constant size their result implies a $\Theta(\sqrt{T})$ regret bound with respect to the best fixed price, which is also proven to be the best possible in that case. In contrast, in the current paper we focus on the case $\hat{\tau} \geq 1$, where the buyers' window sizes may be larger than one, and exhibit the following contributions:

 (i) We present an algorithm that achieves $O(\hat{\tau}^{1/3} T^{2/3})$ additive regret in an adversarial setting, compared to the best fixed posted price in hindsight. The upper bound relies on creating epochs, when the price within each epoch is fixed and the number of epochs limit the number of times the seller switches prices. The actual algorithm that is used to select prices within an epoch is EXP3 (or can be any other multi-arm bandit algorithm with similar performance).

 (ii) We exhibit a matching lower bound of $\Omega(\hat{\tau}^{1/3} T^{2/3})$ regret. The proof of the lower bound reveals that the difficulty in achieving lower regret stems from the lost revenue that the seller suffers every time she tries to lower costs. Buyers from preceding time slots wait and do not purchase the items at the higher prices that prevailed when they arrive. We are thus able to prove a lower bound by reducing to a multi-armed bandit problem with *switching costs*. Our lower bound uses only *two* prices.

In other words, we see that as soon as the buyers' patience increases from zero to one, the optimal regret rate immediately jumps from $\Theta(\sqrt{T})$ to $\Theta(T^{2/3})$.

The rest of the paper is organized as follows. In the remainder of this section we briefly overview related work. We then proceed in Section 2 to provide a formal definition of the model and the statement of our main results. We continue in Section 3 with a presentation of our algorithm and its analysis, present our lower bound in Section 4, and conclude with a brief discussion.

## 1.1  Related work

As mentioned above, the work most closely related to ours is the paper of Kleinberg and Leighton [11] that studies the case $\hat{\tau} = 0$, i.e., in which the buyers' windows are limited to be all of size one. For a fixed set of feasible prices of constant size, their result implies a $\Theta(\sqrt{T})$ regret bound, whereas for a continuum of prices they achieve a $\Theta(T^{2/3})$ regret bound. The $\Omega(T^{2/3})$ lower bound found in [11] is similar to our own in asymptotic magnitude, but stems from the continuous nature of the prices. In our case the lower bound is achieved for buyers with only 2 prices, a case in which Kleinberg and Leighton [11] have a bound of $\Theta(\sqrt{T})$. Hence, we show that such a bound can occur due to the strategic nature of the interaction itself.

A line of work appearing in [1, 12, 13] considers a model of a single buyer and a single seller, where the buyer is strategic and has a constant discount factor. The main issue is that the buyer continuously interacts with the seller and thus has an incentive to lower future prices at the cost of current valuations. They define *strategic regret* and derive near optimal strategic regret bounds for various valuation models. We differ from this line of work in a few important ways. First, they consider other either fixed unknown valuation or stochastic i.i.d. valuations, while we consider adversarial valuations. Second, they consider a single buyer while we consider a stream of buyers. More importantly, in our model the buyers do not influence the prices they are offered, so the strategic incentives are very different. Third, their model uses discounting to model the decay of buyer valuation over time, while we use a window of time.

There is a vast literature in Algorithmic Game Theory on revenue maximization with posted prices, in settings where agents' valuations are drawn from unknown distributions. For the case of a single good of unlimited supply, the goal is to approximate the best price, as a function of the number of samples observed and with a multiplicative approximation ratio. The work of Balcan et al. [4] gives a generic reduction which can be used to show that one can achieve an $\epsilon$-optimal pricing with a sample size $O((H/\epsilon^2) \log(H/\epsilon))$, where $H$ is a bound on the maximum valuation. The works of Cole and Roughgarden [8] and Huang et al. [10] show that for regular and Monotone Hazard Rate distributions sample bounds of $\Theta(\epsilon^{-3})$ and $\Theta(\epsilon^{-3/2})$, respectively, guarantee a multiplicative approximation of $1 - \epsilon$.

Finally, our setting is somewhat similar to a unit-demand auction in which agents desire a single item out of several offerings. In our case, we can consider items sold at different times as different items and agents desire a single one that is within their window. When agents have unit-demand preferences, posted-price mechanisms can extract a constant fraction of the optimal revenue [5, 6, 7]. Note that a constant ratio approximation algorithm implies a linear regret in our model. On the other hand, these works consider a more involved problem from a buyer's valuation perspective.

## 2 Setup and Main Results

We consider a setting with a single seller and a sequence of $T$ buyers $\mathbf{b}_1, \ldots, \mathbf{b}_T$. Every buyer $\mathbf{b}_t$ is associated with *value* $v_t \in [0, 1]$ and *patience* $\tau_t$. A buyer's patience indicates the time duration in which the buyer stays in the system and may purchase an item.

The seller posts prices in advance over some time window. Let $\hat{\tau}$ be the maximum patience, and assume that $\tau_t \leq \hat{\tau}$ for every $t$. Let $p_t$ denote the price at time $t$, and assume that all prices are chosen from a discrete (and normalized) predefined set of $n$ prices $P = \{0, \frac{1}{n}, \frac{2}{n}, \ldots 1\}$. At time $t = 1$, the seller posts prices $p_1, \ldots, p_{\hat{\tau}+1}$, and learns the revenue obtained at time $t = 1$ (the revenue depends on the buyers' behavior, which is explained below). Then, at each time step $t$, the seller publishes a new price $p_{t+\hat{\tau}} \in P$, and learns the revenue obtained at time $t$, which she can use to set the next prices. Note that at every time step, prices are known for the next $\hat{\tau}$ time steps.

The revenue in every time step is determined by the strategic behavior of buyers, which is explained next. Every buyer $\mathbf{b}_t$ observes prices $p_t, \ldots, p_{t+\tau_t}$, and purchases the item at the lowest price among these prices (breaking ties toward earlier times), if she does not exceed her value. The revenue obtained from buyer $\mathbf{b}_t$ is given by:

$$\beta(p_t, \ldots, p_{t+\hat{\tau}}; \mathbf{b}_t) = \begin{cases} \min\{p_t, \ldots, p_{t+\tau_t}\} & \text{if } \min\{p_t, \ldots, p_{t+\tau_t}\} \leq v_t, \\ 0 & \text{otherwise.} \end{cases}$$

As $\mathbf{b}_t$ has patience $\tau_t$, we will sometime omit the irrelevant prices and write $\beta(p_t, \ldots, p_{t+\tau_t}; \mathbf{b}_t) = \beta(p_t, \ldots, p_{t+\hat{\tau}}; \mathbf{b}_t)$.

As we described, a buyer need not buy the item on her day of appearance and may choose to wait. If the buyer chooses to wait, we will observe the feedback from her decision only on the day of purchase. We therefore need to distinguish between the revenue *from buyer t* and the revenue *at time t*. Given a sequence of prices $p_1, \ldots, p_{t+\hat{\tau}}$ and a sequence of buyers $\mathbf{b}_1, \ldots, \mathbf{b}_t$ we define the revenue at time $t$ to be the sum of all revenues from buyers that preferred to buy at time $t$. Formally, let $I_t$ denote the set of all buyers that buy at time $t$, i.e.,

$$I_t = \{\mathbf{b}_i : t = \arg\min\{i \leq t \leq i + \tau_i : p_t = \beta(p_i \ldots, p_{i+\hat{\tau}}; \mathbf{b}_i)\}\}.$$

Then the revenue obtained at time $t$ is given by:

$$R_t(p_{t-\hat{\tau}}, \ldots, p_{t+\hat{\tau}}) = R(p_1, \ldots, p_{t+\hat{\tau}}; \mathbf{b}_{1:t}) := \sum_{i \in I_t} \beta(p_i, \ldots p_{i+\hat{\tau}}; \mathbf{b}_i)),$$

where we use the notation $\mathbf{b}_{1:T}$ as a shorthand for the sequence $\mathbf{b}_1, \ldots, \mathbf{b}_T$. The regret of the (possibly randomized) seller $A$ is the difference between the revenue obtained by the best fixed price in hindsight and the expected revenue obtained by the seller $A$, given a sequence of buyers:

$$\text{Regret}_T(A; \mathbf{b}_{1:T}) = \max_{p^* \in P} \sum_{t=1}^{T} R(p^*, \ldots, p^*; \mathbf{b}_{1:t}) - \mathbb{E}\left[\sum_{t=1}^{T} R(p_1, \ldots p_{t+\hat{\tau}}; \mathbf{b}_{1:t})\right].$$

We further denote by $\text{Regret}_T(A)$ the expected regret a seller $A$ incurs for the worst case sequence, i.e., $\text{Regret}_T(A) = \max_{\mathbf{b}_{1:T}} \text{Regret}_T(A; \mathbf{b}_{1:T})$.

## 2.1 Main Results

Our main result are optimal regret rates in the strategic buyers setting.

**Theorem 1.** *The $T$-round expected regret of Algorithm 1 for any sequence of buyers $\mathbf{b}_1, \ldots, \mathbf{b}_T$ with patience at most $\hat{\tau} \geq 1$ is upper bounded as $\text{Regret}_T \leq 10(\hat{\tau} n \log n)^{1/3} T^{2/3}$.*

**Theorem 2.** *For any $\hat{\tau} \geq 1$, $n \geq 2$ and for any pricing algorithm, there exists a sequence of buyers $\mathbf{b}_1, \ldots, \mathbf{b}_T$ with patience at most $\hat{\tau}$ such that $\text{Regret}_T = \Omega(\hat{\tau}^{1/3} T^{2/3})$.*

## 3 Algorithm

In this section we describe and analyze our online pricing algorithm. It is worth to start by highlighting why simply running an "off the shelf" multi-arm bandit algorithm such as EXP3 would fail. Consider a fixed distribution over the actions and assume the buyer has a window size of two. Unlike the standard multi-arm bandit, where we get the expected revenue from the price we select, now the buyer would select the lower of the two prices, which would clearly hurt our revenue (there is a slight gain, by the increased probability of sell, but it does suffice to offset the loss). For this reason, the seller would intuitively like to minimize the number of time it changes prices (more precisely, lower the prices).

Our online pricing algorithm, which is given in Algorithm 1, is based on the EXP3 algorithm of Auer et al. [3] which we use as a black-box. The algorithm divides the time horizon to roughly $T^{2/3}$ epochs, and within each epoch the seller repeatedly announces the same price, that was chosen by the EXP3 black-box in the beginning of the epoch. In the end of the epoch, EXP3 is updated with the overall average performance of the chosen price during the epoch (ignoring the time steps which might be influenced by different prices). Hence, our algorithm changes the posted price only $O(T^{2/3})$ times, thereby keeping under control the costs associated with price fluctuations due to the patience of the buyers.

---

**Algorithm 1:** Online posted pricing algorithm

---

**Parameters:** horizon $T$, number of prices $n$, and maximal patience $\hat{\tau}$;
Let $B = \lfloor \hat{\tau}^{2/3}(n \log n)^{-1/3} T^{1/3} \rfloor$ and $T' = \lfloor T/B \rfloor$;
Initialize $A \leftarrow \text{EXP3}(T', n)$;
**for** $j = 0, \ldots, T' - 1$ **do**
    Sample $i \sim A$ and let $p'_j = i/n$;
    **for** $t = Bj + 1, \ldots, B(j + 1)$ **do**
        Announce price $p_{t+\hat{\tau}} = p'_j$; %On $j = 0$, $t = 1$ announce $p_1, \ldots p_{t+\tau} = p'_0$.;
        Receive and observe total revenue $R_t(p_{t-\hat{\tau}}, \ldots, p_{t+\hat{\tau}})$;
    Update $A$ with feedback $\frac{1}{B} \sum_{t=Bj+2\hat{\tau}+1}^{B(j+1)} R_t(p_{t-\hat{\tau}}, \ldots, p_{t+\hat{\tau}})$;
**for** $t = BT' + 1, \ldots, T$ **do**
    Announce price $p_{t+\hat{\tau}} = p'_{T'-1}$;

---

We now analyze Algorithm 1 and prove Theorem 1. The proof follows standard arguments in adversarial online learning (e.g., Arora et al. [2]); we note, however, that for obtaining the optimal dependence on the maximal patience $\hat{\tau}$ one cannot apply existing results directly and has to analyse the effect of accumulating revenues over epochs more carefully, as we do in the proof below. This is mainly because in our model the revenue at time $t$ is not bounded by 1 but by $\tau$, hence readily amenable results would add a factor $\tau$ to the regret.

*Proof of Theorem 1.* For all $0 \le j \le T'$ and for all prices $p \in P$, define

$$R'_j(p) = \frac{1}{B} \sum_{t=Bj+2\hat{\tau}+1}^{B(j+1)} R_t(p, \ldots, p).$$

(Here, the argument $p$ is repeated $2\hat{\tau} + 1$ times.) Observe that $0 \le R'_j(p) \le 1$ for all $j$ and $p$, as the maximal total revenue between rounds $Bj + 2\hat{\tau} + 1$ and $B(j + 1)$ is at most $B$; indeed, there are at most $B$ buyers who might make a purchase during that time, and each purchase yields revenue of at most 1. By a similar reasoning, we also have

$$\sum_{t=Bj+1}^{Bj+2\hat{\tau}} R_t(p, \ldots, p) \le 4\hat{\tau} \tag{1}$$

for all $j$ and $p$.

Now, notice that $p_t = p'_j$ for all $Bj + \hat{\tau} + 1 \le t \le B(j + 1) + \hat{\tau}$, hence the feedback fed back to $A$ after epoch $j$ is

$$\frac{1}{B} \sum_{t=Bj+2\hat{\tau}+1}^{B(j+1)} R_t(p_{t-\hat{\tau}}, \ldots, p_{t+\hat{\tau}}) = \frac{1}{B} \sum_{t=Bj+2\hat{\tau}+1}^{B(j+1)} R_t(p'_j, \ldots, p'_j) = R'_j(p'_j).$$

That is, Algorithm 1 is essentially running EXP3 on the reward functions $R'_j$. By the regret bound of EXP3, we know that

$$\sum_{j=0}^{T'-1} R'_j(p^*) - \mathbb{E}\left[ \sum_{j=0}^{T'-1} R'_j(p'_j) \right] \le 3\sqrt{T'n \log n}$$

for any fixed $p^* \in P$, which implies

$$\sum_{j=0}^{T'-1} \sum_{t=Bj+2\hat{\tau}+1}^{B(j+1)} R_t(p^*, \ldots, p^*) - \mathbb{E}\left[ \sum_{j=0}^{T'-1} \sum_{t=Bj+2\hat{\tau}+1}^{B(j+1)} R_t(p_{t-\hat{\tau}}, \ldots, p_{t+\hat{\tau}}) \right] \le 3\sqrt{BTn \log n}. \tag{2}$$

In addition, due to Eq. (1) and the non-negativity of the revenues, we also have

$$\sum_{j=0}^{T'-1} \sum_{t=Bj+1}^{Bj+2\hat{\tau}} R_t(p^*, \ldots, p^*) - \mathbb{E}\left[ \sum_{j=0}^{T'-1} \sum_{t=Bj+1}^{Bj+2\hat{\tau}} R_t(p_{t-\hat{\tau}}, \ldots, p_{t+\hat{\tau}}) \right] \le 4\hat{\tau}T' \le \frac{4\hat{\tau}T}{B}. \tag{3}$$

Summing Eqs. (2) and (3), and taking into account rounds $BT' + 1, \ldots, T$ during which the total revenue is at most $B + 2\hat{\tau}$, we obtain the regret bound

$$\sum_{t=1}^{T} R_t(p^*, \ldots, p^*) - \mathbb{E}\left[ \sum_{t=1}^{T} R_t(p_{t-\hat{\tau}}, \ldots, p_{t+\hat{\tau}}) \right] \le 3\sqrt{BTn \log n} + \frac{4\hat{\tau}T}{B} + B + 2\hat{\tau}.$$

Finally, for $B = \lfloor \hat{\tau}^{2/3}(n \log n)^{-1/3} T^{1/3} \rfloor$, the theorem follows (assuming that $\hat{\tau} < T$). $\qquad \square$

## 4 Lower Bound

We next briefly overview the lower bound and the proof's main technique. A full proof is given in the supplementary material; for simplicity of exposition, here we assume $\hat{\tau} = 1$ and $n = 2$.

Our proof relies on two steps. The first step is a reduction from pricing with patience $\hat{\tau} = 0$ but with switching cost. The second step is to lower bound the regret of pricing with switching cost. This we do again by reduction from the Multi Armed Bandit (MAB) problem with switching cost. We begin by briefly over-viewing these terms and definitions.

We recall the standard setting of MAB with two actions and switching cost $c$. A sequence of losses is produced $\ell_1, \ldots, \ell_T$ where each loss is defined as a function $\ell_t : \{1, 2\} \to \{0, 1\}$. At each round a player chooses an action $i_t \in \{1, 2\}$ and receives as feedback $\ell_t(i_t)$. The switching cost regret of player $A$ is given by

$$S_c\text{-Regret}_T(A; \ell_{1:T}) = \mathbb{E}\left[\sum_{t=1}^{T} \ell_t(i_t) - \min_{i^*} \sum_{t=1}^{T} \ell_t(i^*)\right] + c\mathbb{E}\left[|\{i_t : i_t \neq i_{t-1}\}|\right].$$

We will define analogously the switching cost regret for non-strategic buyers. Namely, given a sequence of buyers $\mathbf{b}_1, \ldots, \mathbf{b}_T$, all with patience $\hat{\tau} = 0$, the switching cost regret for a seller is given by:

$$S_c\text{-Regret}_T(A; \mathbf{b}_{1:T}) = \mathbb{E}\left[\max_{p^*} \sum R(p^*; \mathbf{b}_t) - \sum_{t=1}^{T} R(p_t; \mathbf{b}_t)\right] + c\mathbb{E}\left[|\{p_t : p_t \neq p_{t-1}\}|\right].$$

## 4.1 Reduction from Switching Cost Regret

As we stated above, our first step is to show a reduction from switching cost regret for non-strategic buyers. This we do in Theorem 3:

**Theorem 3.** *For every (possibly randomized) seller $A$ for strategic buyers with patience at most $\hat{\tau} = 1$, there exists a randomized seller $A'$ for non-strategic buyers with patience $\hat{\tau} = 0$ such that:*

$$\tfrac{1}{2}S_{\frac{1}{12}}\text{-Regret}_T(A') \leq \text{Regret}_T(A)$$

The proof idea is to construct from every sequence of non-strategic buyers $\mathbf{b}_1, \ldots, \mathbf{b}_T$ a sequence of strategic buyers $\bar{\mathbf{b}}_1, \ldots, \bar{\mathbf{b}}_T$ such that the regret incurred to $A$ by $\bar{\mathbf{b}}_{1:T}$ is at least the switching cost regret incurred to $A'$ by $\mathbf{b}_{1:T}$. The idea behind the construction is as follows: At each iteration $t$ we choose with probability half to present to the seller $\mathbf{b}_t$ and with probability half we present to the seller a buyer $\mathbf{z}_t$ that has the following statistics:

$$\mathbf{z}_t = \begin{cases} (v = \tfrac{1}{2}, \tau = 0) & \text{w.p. } \tfrac{1}{2} \\ (v = 1, \tau = 1) & \text{w.p. } \tfrac{1}{2} \end{cases} \tag{4}$$

That is, $\mathbf{z}_t$ is with probability $\tfrac{1}{2}$ a buyer with value $v = \tfrac{1}{2}$ and patience $\tau = 0$, and with probability $\tfrac{1}{2}$, $\mathbf{z}_t$ is a buyer with value $v = 1$ and patience $\tau = 1$.

Observe that if $\mathbf{z}_t$ would always have patience $\tau = 0$ (i.e., even if her value is $v = 1$), for any sequence of prices the expected rewards from the $\mathbf{z}_t$ buyer is always half, independent of the prices. In other words, the sequence of noise does not change the performance of the sequence of prices and cannot be exploited to improve. On the other hand, note since the value 1 corresponds to patience 1, the seller might lose half whenever she reduces the price from 1 to $\tfrac{1}{2}$. A crucial point is that the seller must post her price in advance, therefore she cannot in any way predict if the buyer is willing to wait or not and manipulate prices accordingly. A proof for the following Lemma is provided in the supplementary material.

**Lemma 4.** *Consider the pricing problem with $\hat{\tau} = 1$ and $n = 2$. Let $\mathbf{b}_1, \ldots, \mathbf{b}_T$ be a sequence of buyers with patience 0. Let $\mathbf{z}_1, \ldots, \mathbf{z}_T$ be a sequence of stochastic buyers as in Eq. (4). Define $\bar{\mathbf{b}}_t$ to be a stochastic buyer that is with probability half $\mathbf{b}_t$ and with probability half $\mathbf{z}_t$. Then, for any seller $A$, the expected regret $A$ incurs from the sequence $\bar{\mathbf{b}}_{1:T}$ is at least*

$$\mathbb{E}\left[\text{Regret}_T(A; \bar{\mathbf{b}}_{1:T})\right] \geq \frac{1}{2}\mathbb{E}\left[\max_{p^* \in P} \sum_{t=1}^{T} \beta(p^*; \mathbf{b}_t) - \beta(p_t; \mathbf{b}_t)\right] + \frac{1}{8}\mathbb{E}\left[\sum_{t=1}^{T} |\{p_t : p_t > p_{t+1}\}|\right] \tag{5}$$

*where the expectations are taken with respect to the internal randomization of the seller $A$ and the random bits used to generate the sequence $\bar{\mathbf{b}}_{1:T}$.*

### 4.1.1 Proof for Theorem 3

To construct algorithm $A'$ from $A$, we develop a meta algorithm $\mathcal{A}$, depicted in Algorithm 2 that receives an algorithm, or seller, as input. $A'$ is then the seller obtained by fixing $A$ as the input for $\mathcal{A}$. In our reduction we assume that at each iteration algorithm $\mathcal{A}$ can ask from $A$ one posted price, $p_t$, and in turn she can return a feedback $r_t$ to algorithm $A$, then a new iteration begins.

The idea of construction is as follows: As an initialization step Algorithm $A'$ produces a stochastic sequence of buyers of type $\mathbf{z}_1, \ldots, \mathbf{z}_t$, the algorithm then chooses apriori if at step $t$ a buyer $\bar{\mathbf{b}}_t$ is going to be the buyer $\mathbf{b}_t$ that she observes or $\mathbf{z}_t$ (with probability half each). The sequence $\bar{\mathbf{b}}_t$ is distributed as depicted in Lemma 4. Note that we do not assume that the learner knows the value of $\mathbf{b}_t$.

At each iteration $t$, algorithm $A'$ receives price $p_t$ from algorithm $A$ and posts price $p_t$. She then receives as feedback $\beta(p_t; \mathbf{b}_t)$: Given the revenues $\beta(p_1; \mathbf{b}_1), \ldots, \beta(p_t; \mathbf{b}_t)$ and her own internal random variables, the algorithm can calculate the revenue for algorithm $A$ w.r.t to the sequence of buyers $\bar{\mathbf{b}}_1, \ldots, \bar{\mathbf{b}}_t$, namely $r_t = R(p_{t-1}, \ldots, p_{t+1}, \bar{\mathbf{b}}_{1:t})$.

In turn, at time $t$ algorithm $A'$ returns to algorithm $A$ her revenue, or feedback, w.r.t $\bar{\mathbf{b}}_1, \ldots, \bar{\mathbf{b}}_T$ at time $t$ which is $r_t$.

Since Algorithm $A$ receives as feedback at time $t$ $R(p_{t-1}, p_t, p_{t+1}; \bar{\mathbf{b}}_{1:t})$, we obtain that for the sequence of posted prices $p_1, \ldots, p_T$:

$$\text{Regret}_T(A; \bar{\mathbf{b}}_{1:T}) = \sum_{t=1}^{T} \beta(p^*, p^*; \bar{\mathbf{b}}_t) - \sum_{t=1}^{T} \beta(p_t, p_{t+1}; \bar{\mathbf{b}}_t).$$

Taking expectation, using Lemma 4, and noting that the number of time $p_{t+1} > p_t$ is at least $1/3$ of the times $p_t \neq p_{t+1}$ (since there are only 2 prices), we have that

$$\frac{1}{2} \text{S}_{\frac{1}{12}}\text{-Regret}_T(A'; \mathbf{b}_{1:T}) \le \mathbb{E}_{\bar{\mathbf{b}}_{1:T}} \left[ \text{Regret}_T(A; \bar{\mathbf{b}}_{1:T}) \right] \le \text{Regret}_T(A)$$

Since this is true for *any* sequence $\mathbf{b}_{1:T}$ we obtain the desired result.

---

**Algorithm 2:** Reduction from from pricing with switching cost to strategic buyers

---

**Input**:$T$, $A$ % A is an algorithm with bounded regret for strategic buyers;
**Output**:$p_1, \ldots, p_T$;
Set $r_1 = \ldots = r_T = 0$;
Draw IID $\mathbf{z}_1, \ldots, \mathbf{z}_T$ % see Eq. 4;
Draw IID $e_1, \ldots, e_T \in \{0, 1\}$ Distributed according to Bernoulli distribution;
**for** *t=1,…,T* **do**
    Receive from $A$ a posted price $p_{t+1}$; %At first round receive two prices $p_1, p_2$.;
    post price $p_t$ and receive as feedback $\beta(p_t; \mathbf{b}_t)$;
    **if** $e_t = 0$ **then**
        Set $r_t = r_t + \beta(p_t; \mathbf{b}_t)$; % $\bar{\mathbf{b}}_t = \mathbf{b}_t$
    **else**
        **if** $(p_t \le p_{t+1})$ *OR* $(\mathbf{z}_t$ *has patience 0)* **then**
            Set $r_t = r_t + \beta(p_t; \mathbf{z}_t)$
        **else**
            Set $r_{t+1} = r_{t+1} + \beta(p_t, p_{t+1}; \mathbf{z}_t)$
    Return $r_t$ as feedback to $A$.

---

### 4.2 From MAB with switching cost to Pricing with switching cost

The above section concluded that switching cost for pricing may be reduced to pricing with strategic buyers. Therefore, our next step would be to show that we can produce a sequence of non-strategic buyers with high switching cost regret. Our proof relies on a further reduction for MAB with Switching cost.

**Theorem 5** (Dekel et al. [9]). *Consider the MAB setting with 2 actions. For any randomized player, there exists a sequence of loss functions $\ell_1, \ldots, \ell_T$ where $\ell_t : \{1, 2\} \to \{0, 1\}$ such that $S_c\text{-Regret}_T(A; \ell_{1:T}) \in \Omega(T^{2/3})$, for every $c > 0$.*

Here we prove an analogous statement for pricing setting:

**Theorem 6.** *Consider the pricing problem for buyers with patience $\hat{\tau} = 0$ and $n = 2$. For any randomized seller, there exists a sequence of buyers $\mathbf{b}_1, \ldots, \mathbf{b}_T$ such that $S_c\text{-}Regret_T(A; \mathbf{b}_{1:T}) \in \Omega(T^{2/3})$, for every $c > 0$.*

The transition from MAB with switching cost to pricing with switching cost is a non-trivial task. To do so, we have to relate actions to prices and values to loss vectors in a manner that would relate the revenue regret to the loss regret. The main challenge, perhaps, is that the structure of the feedback is inherently different in the two problems. In two-armed bandit problems all loss configuration are feasible. In contrast, in the pricing case certain feedbacks collapse to full information: for example, if we sell at price 1 we know the feedback from price $\frac{1}{2}$, and if we fail to sell at price $\frac{1}{2}$ we obtain full feedback for price 1.

Our reduction proceeds roughly along the following lines. We begin by constructing stochastic mappings that turn loss vectors into values $v_t : \{0, 1\}^2 \to \{0, \frac{1}{2}, 1\}$. This in turn defines a mapping from a sequences of losses $\ell_t$ to stochastic sequences of buyers $\mathbf{b}_t$. In our reduction we assume we are given an algorithm $A$ that solves the pricing problem; that is, at each iteration we may ask for a price and then in turn we return a feedback $\beta(p_t; \mathbf{b}_t)$. Note that we cannot assume that we have access or know $\mathbf{b}_t$ that is defined by $v_t(\ell_t)$. The buyer $\mathbf{b}_t$ depends on the full loss vector $\ell_t$: assuming that we can see the full $\ell_t$ would not lead to a meaningful reduction for MAB.

However, our construction of $v_t$ is such that each posted price is associated with a single action. This means that for each posted price there is a single action we need to observe in order to calculate the correct feedback or revenue. This also means that we switch actions only when algorithm $A$ switches prices. Finally, our sequence of transformation has the following property: if $i$ is the action needed in order to discover the revenue for price $p$, then $\mathbb{E}(\ell_t(i)) = \frac{1}{2} - \frac{1}{4}\mathbb{E}(\beta(p; \mathbf{b}_t))$. Thus, the regret for our actions compares to the regret of the seller.

## 5  Discussion

In this work we introduced a new model of strategic buyers, where buyers have a window of time in which they would like to purchase the item. Our modeling circumvents complicated dynamics between the buyers, since it forces the seller to post prices for the entire window of time in advance.

We consider an adversarial setting, where both buyer valuation and window size are selected adversarially. We compare our online algorithm to a static fixed price, which is by definition oblivious to the window sizes. We show that the regret is sub-linear, and more precisely $\Theta(T^{2/3})$. The upper bound shows that in this model the average regret per buyer is still vanishing. The lower bound shows that having a window size greater than 1 impacts the regret bounds dramatically. Even for window sizes 1 or 2 and prices $\frac{1}{2}$ or 1 we get a regret of $\Omega(T^{2/3})$, compared to a regret of $O(T^{1/2})$ when all the windows are of size 1.

Given the sharp $\Theta(T^{2/3})$ bound, it might be worth revisiting our feedback model. Our model assumes that the feedback for the seller is the revenue obtained at the end of each day. It is worthwhile to consider stronger feedback models, where the seller can gain more information about the buyers. Namely, their day of arrival and their window size. In terms of the upper bound, our result applies to any feedback model that is stronger, i.e., as long as the seller gets to observe the revenue per day, the $O(T^{2/3})$ bound holds. As far as the lower bound is concerned, one can observe that our proofs and construction are valid even for very strong feedback models. Namely, even if the seller gets as feedback the revenue from buyer $t$ at time $t$ (instead of the time of purchase), and in fact even if she gets to observe the patience of the buyers (i.e. full information w.r.t. patience), the $\Omega(T^{2/3})$ bound holds, as long as the seller posts prices in advance.

We did not consider continuous pricing explicitly, but one can verify that applying our algorithm to a setting of continuous pricing gives a regret bound of $O(T^{3/4})$, by discretizing the continuous prices to $T^{1/4}$ prices. On the positive side, it shows that we still obtain a vanishing average regret in the continuous case. On the other hand, we were not able to improve our lower bound to match this upper bound. This gap is one of the interesting open problems in our work.

# References

[1] K. Amin, A. Rostamizadeh, and U. Syed. Learning prices for repeated auctions with strategic buyers. In C. J. C. Burges, L. Bottou, M. Welling, Z. Ghahramani, and K. Q. Weinberger, editors, *Advances in Neural Information Processing Systems 26*, pages 1169–1177. 2013.

[2] R. Arora, O. Dekel, and A. Tewari. Online bandit learning against an adaptive adversary: from regret to policy regret. *arXiv preprint arXiv:1206.6400*, 2012.

[3] P. Auer, N. Cesa-Bianchi, Y. Freund, and R. E. Schapire. The nonstochastic multiarmed bandit problem. *SIAM Journal on Computing*, 32(1):48–77, 2002.

[4] M.-F. Balcan, A. Blum, J. D. Hartline, and Y. Mansour. Reducing mechanism design to algorithm design via machine learning. *J. Comput. Syst. Sci.*, 74(8):1245–1270, 2008.

[5] S. Chawla, J. D. Hartline, and R. D. Kleinberg. Algorithmic pricing via virtual valuations. In *ACM Conference on Electronic Commerce*, pages 243–251, 2007.

[6] S. Chawla, J. D. Hartline, D. L. Malec, and B. Sivan. Multi-parameter mechanism design and sequential posted pricing. In *STOC*, pages 311–320, 2010.

[7] S. Chawla, D. L. Malec, and B. Sivan. The Power of Randomness in Bayesian Optimal Mechanism Design. In *the 11th ACM Conference on Electronic Commerce (EC)*, 2010.

[8] R. Cole and T. Roughgarden. The sample complexity of revenue maximization. In *Proceedings of the 46th Annual ACM Symposium on Theory of Computing*, pages 243–252. ACM, 2014.

[9] O. Dekel, J. Ding, T. Koren, and Y. Peres. Bandits with switching costs: T 2/3 regret. In *Proceedings of the 46th Annual ACM Symposium on Theory of Computing*, pages 459–467. ACM, 2014.

[10] Z. Huang, Y. Mansour, and T. Roughgarden. Making the most of your samples. In *Proceedings of the Sixteenth ACM Conference on Economics and Computation, EC*, pages 45–60, 2015.

[11] R. D. Kleinberg and F. T. Leighton. The value of knowing a demand curve: Bounds on regret for online posted-price auctions. In *44th Symposium on Foundations of Computer Science FOCS*, pages 594–605, 2003.

[12] M. Mohri and A. Munoz. Optimal regret minimization in posted-price auctions with strategic buyers. In Z. Ghahramani, M. Welling, C. Cortes, N. D. Lawrence, and K. Q. Weinberger, editors, *Advances in Neural Information Processing Systems 27*, pages 1871–1879. 2014.

[13] M. Mohri and A. Munoz. Revenue optimization against strategic buyers. In C. Cortes, N. D. Lawrence, D. D. Lee, M. Sugiyama, and R. Garnett, editors, *Advances in Neural Information Processing Systems 28*, pages 2530–2538. 2015.

